# A General Greedy Approximation Algorithm with Applications

**Tong Zhang**
IBM T.J. Watson Research Center
Yorktown Heights, NY 10598
tzhang@watson.ibm.com

## Abstract

Greedy approximation algorithms have been frequently used to obtain sparse solutions to learning problems. In this paper, we present a general greedy algorithm for solving a class of convex optimization problems. We derive a bound on the rate of approximation for this algorithm, and show that our algorithm includes a number of earlier studies as special cases.

## 1 Introduction

The goal of machine learning is to obtain a certain input/output functional relationship from a set of training examples. In order to do so, we need to start with a model of the functional relationship. In practice, it is often desirable to find the simplest model that can explain the data. This is because simple models are often easier to understand and can have significant computational advantages over more complicated models. In addition, the philosophy of Occam's Razor implies that the simplest solution is likely to be the best solution among all possible solutions,

In this paper, we are interested in composite models that can be expressed as linear combinations of basic models. In this framework, it is natural to measure the simplicity of a composite model by the number of its basic model components. Since a composite model in our framework corresponds to a linear weight over the basic model space, therefore our measurement of model simplicity corresponds to the sparsity of the linear weight representation.

In this paper, we are interested in achieving sparsity through a greedy optimization algorithm which we propose in the next section. This algorithm is closely related to a number of previous works. The basic idea was originated in [5], where Jones observed that if a target vector in a Hilbert space is a convex combination of a library of basic vectors, then using greedy approximation, one can achieve an error rate of $O(1/k)$ with $k$ basic library vectors. The idea has been refined in [1] to analyze the approximation property of sigmoidal functions including neural networks.

The above methods can be regarded as greedy sparse algorithms for functional approximation, which is the noise-free case of regression problems. A similar greedy algorithm can also be used to solve general regression problems under noisy conditions [6]. In addition to regression, greedy approximation can also be applied to classification problems.

The resulting algorithm is closely related to boosting [2] under the additive model point of view [3]. This paper shows how to generalize the method in [5, 1] for analyzing greedy algorithms (in their case, for functional approximation problems) and apply it to boosting. Detailed analysis will be given in Section 4. Our method can also be used to obtain sparse kernel representations for regression problems. Such a sparse representation is what support vector regression machines try to achieve. In this regard, the method given in this paper complements some recently proposed greedy kernel methods for Gaussian processes such as [9, 10].

The proposed greedy approximation method can also be applied to other prediction problems with different loss functions. For example, in density estimation, the goal is to find a model that has the smallest negative log-likelihood. A greedy algorithm was analyzed in [7]. Similar approximation bounds can be directly obtained under the general framework proposed in this paper.

We proceed as follows. Section 2 formalizes the general class of problems considered in this paper, and proposes a greedy algorithm to solve the formulation. The convergence rate of the algorithm is investigated in Section 3. Section 4 includes a few examples that can be obtained from our algorithm. Some final concluding remarks are given in Section 5.

## 2    General Algorithm

In machine learning, our goal is often to predict an unobserved output value $y$ based on an observed input vector $x$. This requires us to estimate a functional relationship $y \approx p(x)$ from a set of example pairs of $(x, y)$. Usually the quality of the predictor $p(x)$ can be measured by a loss function $L(p(x), y)$ that is problem dependent.

In this paper, we are interested in the following scenario: given a family of basic predictors $p(\theta, x)$ parameterized by $\theta$, we want to obtain a good predictor $p(x)$ that lies in the convex hull of $p(\theta, x)$ with the fewest possible terms: $p(x) = \sum_{i=1}^{k} w_i p(\theta_i, x)$, where $w_i$ are non-negative weights so that $\sum_{i=1}^{k} w_i = 1$. This family of models can be regarded as additive models in statistics [4]. Formally, each basic model $p(\theta, x)$ can be regarded as a vector in a linear functional space. Our problem in its most general form can thus be described as to find a vector $p(x)$ in the convex hull of $p(\theta, x)$ to minimize a functional $f$ of $p$ that measures the quality of $p$. This functional $f$ of $p$ plays the role of loss function for learning problems.

More formally, we consider a linear vector space $V$, and a subset $S \in V$. Denote by $\mathrm{co}(S)$ the convex hull of $S$:

$$\mathrm{co}(S) = \{\sum_{j=1}^{m} \alpha_j u_j : \alpha_j \geq 0, \sum_{j=1}^{m} \alpha_j = 1, u_j \in S, m \in Z^+\},$$

where we use $Z^+$ to denote the set of positive integers.

We consider the following optimization problem on $\mathrm{co}(S)$:

$$\inf_{v \in \mathrm{co}(S)} f(v). \tag{1}$$

In this paper, we assume that $f$ is a differentiable convex function on $\mathrm{co}(S)$.

We propose the following algorithm to approximately solved (1).

**Algorithm 2.1**  *(Sparse greedy approximation)*

```
given v^0 ∈ co(S)
for k = 1, 2, . . .
    find v̄_k ∈ S and 0 ≤ α_k ≤ 1 that minimize
        f((1 − α_k)v^{k−1} + α_k v̄_k)      (∗)
    let v^k = (1 − α_k)v^{k−1} + α_k v̄_k
end
```

For simplicity, we assume that the minimization of $(*)$ in Algorithm 2.1 can be exactly achieved at each step. This assumption is not essential, and can be easily removed using a slightly more refined analysis. However due to the space limitation, we shall not consider this generalization.

For convenience, we introduce the following quantity

$$\Delta f(v) = f(v) - \inf_{v' \in \mathrm{co}(S)} f(v').$$

In the next section, we show that under appropriate regularity conditions, $\Delta f(v^k) \to 0$ as $k \to +\infty$, where $v^k$ is computed from Algorithm 2.1. In addition, the convergence rate can be bounded as $O(1/k)$.

## 3   Approximation bound

Given any convex function $f$, we have the following proposition, which is a direct consequence of the definition of convexity. In convex analysis, The gradient $\nabla f$ can be replaced by the concept of *subgradient*, which we do not consider in this paper for simplicity.

**Proposition 3.1** *Consider a convex function $f(v)$, and two vectors $v$ and $v'$, we have*

$$f(v') - f(v) \geq (v' - v)^T \nabla f(v),$$

*where $\nabla f$ is the gradient of $f$.*

The following lemma is the main theoretical result of the paper, which bounds the performance of each greedy solution step in Algorithm 2.1. We assume that $f$ is second order differentiable.

**Lemma 3.1** *Let*

$$M = \sup_{v,v' \in \mathrm{co}(S)} v^T \nabla^2 f(v')v,$$

*where we assume that the Hessian $\nabla^2 f$ of $f$ exists everywhere in $\mathrm{co}(S)$. For all vectors $v \in \mathrm{co}(S)$: if $\Delta f(v) \geq 4M$, we have*

$$\inf_{\eta \in [0,1], v' \in S} \Delta f((1-\eta)v + \eta v') \leq 2M;$$

*if $\Delta f(v) \leq 4M$, we have*

$$\inf_{\eta \in [0,1], v' \in S} \Delta f((1-\eta)v + \eta v') \leq \Delta f(v) - \frac{\Delta f(v)^2}{8M}.$$

*Proof.* Using Taylor expansion and the definition of $M$, we have the following inequality for all $v \in \mathrm{co}(S)$, $v' \in S$, and $\eta \in [0,1]$,

$$f((1-\eta)v + \eta v') - f(v) \leq \eta(v' - v)^T \nabla f(v) + \frac{\eta^2}{2}(4M).$$

Now, consider two sequences $\alpha_j \geq 0$ and $v'_j \in S$ $(j = 1, \ldots, m)$, such that $\sum_{j=1}^m \alpha_j = 1$. Multiply the above inequality (with $v'$ replaced by $v'_j$) by $\alpha_j$, and sum over $j$, we obtain

$$\sum_{j=1}^m \alpha_j f((1-\eta)v + \eta v'_j) - f(v) \leq \eta(\sum_{j=1}^m \alpha_j v'_j - v)^T \nabla f(v) + 2\eta^2 M.$$

It is easy to see that this implies the inequality

$$\inf_j f((1-\eta)v + \eta v'_j) - f(v) \leq \eta(\sum_{j=1}^m \alpha_j v'_j - v)^T \nabla f(v) + 2\eta^2 M.$$

Using Proposition 3.1, we obtain

$$\inf_j f((1-\eta)v + \eta v'_j) - f(v) \leq \eta(f(\sum_{j=1}^m \alpha_j v'_j) - f(v)) + 2\eta^2 M.$$

Since in the above, $\alpha_j$ and $v'_j$ are arbitrary, therefore $\sum_{j=1}^m \alpha_j v'_j$ can be used to express any vector $v \in co(S)$. This implies

$$\inf_{v' \in S} f((1-\eta)v + \eta v') \leq f(v) + \eta(\inf_{\bar{v}} f(\bar{v}) - f(v)) + 2\eta^2 M.$$

Now by setting $\eta = \min(1, (f(v) - \inf_{\bar{v}} f(\bar{v}))/(4M))$ in the above inequality, we obtain the lemma. $\square$

Using the above lemma and note that $\Delta f(v^1) \leq 2M$, it is easy to obtain the following theorem by induction. For space limitation, we skip the proof.

**Theorem 3.1** *Under the assumptions of Lemma 3.1, Algorithm 2.1 approximately solves (1), and the rate of convergence for $k \geq 1$ is given by*

$$\Delta f(v^k) \leq \frac{8M}{k+3}.$$

*If $\Delta f(v^0) \leq 4M$, then we also have*

$$\Delta f(v^k) \leq \frac{8M}{k + \frac{8M}{\Delta f(v^0)}}.$$

# 4   Examples

In this section, we discuss the application of Algorithm 2.1 in some learning problems. We show that the general formulation considered in this paper includes some previous formulations as special cases. We will also compare our results with similar results in the literature.

## 4.1   Regression

In regression, we would like to approximate $y$ as $p(x)$ so that the expected loss of

$$f(p(\cdot)) = E_{x,y}(y - p(x))^2$$

is small, where we use the squared loss for simplicity (this choice is obviously not crucial in our framework). $E_{x,y}$ is the expectation over $x$ and $y$, which often corresponds to the empirical distribution of $(x,y)$ pairs. It may also represent the true distribution for some

other engineering applications. Given a set of basis functions $p(\theta, x)$ with $\theta \in \Omega$, we may consider the following regression formulation that is slightly different from (1):

$$\inf_w E_{x,y}(y - \sum_{i=1}^{k} w_i p(\theta_i, x))^2 \tag{2}$$

$$\text{s.t.} \quad \sum_{i=1}^{k} |w_i| \leq A,$$

where $A$ is a positive regularization parameter which is used to control the size of the weight vector $w$. The above formulation can be readily converted into (1) by considering the following set $S$ of basic vectors:

$$S = \{ap(\theta, x) : |a| \leq A, \theta \in \Omega\}.$$

We may start with $w^0 = 0$ ($v^0 = 0$) in Algorithm 2.1. Since the quantity $M$ in Lemma 3.1 can be bounded as

$$M = \sup_\theta 2A^2 E_x p(\theta, x)^2.$$

This implies that the sparse solution $v^k$ in Algorithm 2.1, represented as weight $\|w^k\| \leq A$ and $\theta_i$ ($i = 1, \ldots, k$), satisfies the following inequality:

$$E_{x,y}(y - \sum_{i=1}^{k} w_i^k p(\theta_i, x))^2 \leq \inf_{\|w\|_1 \leq A, \theta_j'} E_{x,y}(y - \sum_{j=1}^{m} w_j p(\theta_j', x))^2 + \frac{16A^2 \sup_\theta E_x p(\theta, x)^2}{k+3}$$

for all $k \geq 1$. This leads to the original functional approximation results in [1, 5] and its generalization in [6].

The sparse regression algorithm studied in this section can also be applied to kernel methods. In this case, $\Omega$ corresponds to the input training data space $\{x_1, \ldots, x_n\}$, and the basis predictors are of the form $p(\theta, x) = k(x_i, x)$. Clearly, this corresponds to a special case of (2). A sparse kernel representation can be obtained easily from Algorithm 2.1 which leads to provably good approximation rate. Our sparse kernel regression formulation is related to Gaussian processes, where greedy style algorithms have also been proposed [9, 10]. The bound given here is comparable to the bound given in [10] where a sparse approximation rate of the form $O(1/k)$ was obtained.

## 4.2 Binary classification and Boosting

In binary classification, the output value $y \in \{\pm 1\}$ is a discrete variable. Given a continuous model $p(x)$, we consider the following prediction rule:

$$y = \begin{cases} 1 & \text{if } p(x) \geq 0, \\ -1 & \text{if } p(x) < 0. \end{cases}$$

The classification error (we shall ignore the point $p(x) = 0$, which is assumed to occur rarely) can be given by

$$L(p(x), y) = \begin{cases} 1 & \text{if } p(x)y \leq 0, \\ 0 & \text{if } p(x)y > 0. \end{cases}$$

Unfortunately, this classification error function is not convex, which cannot be handled in our formulation. In fact, even in many other popular methods, such as logistic regression and support vector machines, some kind of convex formulations have to be employed.

Although it is possible for us to analyze their formulations, in this section, we only consider the following form of loss that is closely related to Adaboost [2]:

$$f(p(\cdot)) = \ln(E_{x,y} \exp(-Ap(x)y)), \tag{3}$$

where $A$ is a scaling factor.

Again, we consider a set of basis predictors $p(\theta, x) \in [-1, 1]$, which are often called weak learners in the boosting literature. We would like to find a strong learner $p(x)$ as a convex combination of weak learners to approximately minimize the above loss:

$$\inf_w \ln(E_{x,y} \exp(-A \sum_{i=1}^{k} w_i p(\theta_i, x)y)) \tag{4}$$

$$\text{s.t.} \quad \sum_{i=1}^{k} w_i \leq 1, \quad w_i \geq 0. \tag{5}$$

This can be written as formulation (1) with

$$S = \{ap(\theta, x) : 0 \leq a \leq 1\}.$$

Using simple algebra, it is easy to verify that

$$M \leq \sup_{p, p' \in \text{co}(S)} \frac{A^2 E_{x,y} (\exp(-Ap'(x)y)p(x)^2)}{E_{x,y} \exp(-Ap'(x)y)} \leq A^2.$$

We start with $w^0 = 0$ in Algorithm 2.1. Theorem 3.1 implies that the sparse solution $v^k$, represented as weight $w^k$ and $\theta_i$ $(i = 1, \ldots, k)$, satisfies the following inequality:

$$E_{x,y} \exp(-A \sum_{i=1}^{k} w_i^k p(\theta_i, x)y) \leq \inf_{\|w\|_1 \leq 1, \theta_j'} E_{x,y} \exp(-A \sum_{j=1}^{m} w_j p(\theta_j', x)y + \frac{8A^2}{k+3}) \tag{6}$$

for all $k \geq 1$. Weight $w$ in the above inequality is non-negative. Now we consider the special situation that there exists $\gamma > 0$ such that

$$\inf_{\|w\|_1 \leq 1, \theta_j'} E_{x,y} \exp(-A \sum_{j=1}^{m} w_j p(\theta_j', x)y) \leq \exp(-2\gamma A). \tag{7}$$

This condition will be satisfied in the large margin linearly separable case where there exists $w_j \geq 0, \theta_j'$ and $\gamma > 0$ such that $\|w\|_1 \leq 1$ and for all data $(x, y)$,

$$\sum_{j=1}^{m} w_j p(\theta_j', x)y \geq 2\gamma.$$

Now, under (7), we obtain from (6) that

$$P(\sum_{i=1}^{k} w_i^k p(\theta_i, x)y \leq \gamma) \leq \exp(-A\gamma + \frac{8A^2}{k+3}).$$

Fix any $k \geq 1$, we can choose $A = \gamma(k+3)/16$ to obtain

$$P(\sum_{i=1}^{k} w_i^k p(\theta_i, x)y \leq \gamma) \leq \exp(-\gamma^2(k+3)/32). \tag{8}$$

This implies that the misclassification error rate decays exponentially. The exponential decay of misclassification error is the original motivation of Adaboost [2]. Boosting was later

viewed as greedy approximation in the additive model framework [3]. From the learning theory perspective, the good generalization ability of boosting is related to its tendency to improve the misclassification error under a positive margin [8]. From this point of view, inequality (8) gives a much more explicit margin error bound (which decreases exponentially) than a related result in [8].

In the framework of additive models, Adaboost corresponds to the exponential loss (3) analyzed in this section. As pointed out in [3], other loss functions can also be used. Using our analysis, we may also obtain sparse approximation bounds for these different loss functions. However, it is also easy to observe that they will not lead to the exponential decay of classification error in the separable case. Although the exponential loss in (3) is attractive for separable problems due to the exponential decay of margin error, it is very sensitive to outliers in the non-separable case.

We shall mention that an interesting aspect of boosting is the concept of adaptive resampling or sample reweighting. Although this idea has dominated the interpretation of boosting algorithms, it has been argued in [3] that adaptive resampling is only a computational by-product. The idea corresponds to a Newton step approximation in the sparse greedy solution of $(*)$ in Algorithm 2.1 under the additive model framework which we consider here. Our analysis further confirmed that the greedy sparse solution of an additive model in (1), rather than reweighting itself is the key component in boosting. In our framework, it is also much easier to related the idea of boosting to the greedy function approximation method outlined in [1, 5].

### 4.3 Mixture density estimation

In mixture density estimation, the output $y$ is the probability density function of the input vector at $x$. The following negative log-likelihood is commonly used as loss function:

$$f(p(\cdot)) = -E_x \ln p(x),$$

where $p(x) \geq 0$ is a probability density function.

Again, we consider a set of basis predictors $p(\theta, x)$, which are often called mixture components. We would like to find a mixture probability density model $p(x)$ as a convex combination of mixture components to approximately minimize the negative log-likelihood:

$$\inf_w -E_x \ln(\sum_{i=1}^k w_i p(\theta_i, x)y)) \tag{9}$$

$$\text{s.t.} \quad \sum_{i=1}^k w_i = 1, \quad w_i \geq 0. \tag{10}$$

This problem was studied in [7]. The quantity $M$ defined in Lemma 3.1 can be computed as:

$$M = \sup_{q_1(\cdot), q_2(\cdot) \in \mathrm{co}(S)} E_x \frac{q_1(x)^2}{q_2(x)^2} = \sup_{\theta_1, \theta_2} E_x \frac{p(\theta_1, x)^2}{p(\theta_2, x)^2}.$$

An approximation bound can now be directly obtained from Theorem 3.1. It has a form similar to the bound given in [7].

## 5    Conclusion

This paper studies a formalization of a general class of prediction problems in machine learning, where the goal is to approximate the best model as a convex combination of

a family of basic models. The quality of the approximation can be measured by a loss function which we want to minimize.

We proposed a greedy algorithm to solve the problem, and we have shown that for a variety of loss functions, a convergence rate of $O(1/k)$ can be achieved using a convex combination of $k$ basic models. We have illustrated the consequence of this general algorithm in regression, classification and density estimation, and related the resulting algorithms to previous methods.

# References

[1] A.R. Barron. Universal approximation bounds for superpositions of a sigmoidal function. *IEEE Transactions on Information Theory*, 39(3):930–945, 1993.

[2] Y. Freund and R.E. Schapire. A decision-theoretic generalization of on-line learning and an application to boosting. *J. Comput. Syst. Sci.*, 55(1):119–139, 1997.

[3] Jerome Friedman, Trevor Hastie, and Robert Tibshirani. Additive logistic regression: A statistical view of boosting. *The Annals of Statistics*, 28(2):337–407, 2000. With discussion.

[4] T. J. Hastie and R. J. Tibshirani. *Generalized additive models*. Chapman and Hall Ltd., London, 1990.

[5] Lee K. Jones. A simple lemma on greedy approximation in Hilbert space and convergence rates for projection pursuit regression and neural network training. *Ann. Statist.*, 20(1):608–613, 1992.

[6] Wee Sun Lee, P.L. Bartlett, and R.C. Williamson. Efficient agnostic learning of neural networks with bounded fan-in. *IEEE Transactions on Information Theory*, 42(6):2118–2132, 1996.

[7] Jonathan Q. Li and Andrew R. Barron. Mixture density estimation. In S.A. Solla, T.K. Leen, and K.-R. Müller, editors, *Advances in Neural Information Processing Systems 12*, pages 279–285. MIT Press, 2000.

[8] Robert E. Schapire, Yoav Freund, Peter Bartlett, and Wee Sun Lee. Boosting the margin: a new explanation for the effectiveness of voting methods. *Ann. Statist.*, 26(5):1651–1686, 1998.

[9] Alex J. Smola and Peter Bartlett. Sparse greedy Gaussian process regression. In *Advances in Neural Information Processing Systems 13*, pages 619–625, 2001.

[10] Tong Zhang. Some sparse approximation bounds for regression problems. In *The Eighteenth International Conference on Machine Learning*, pages 624–631, 2001.
